# A Cost Function for Internal Representations

**Anders Krogh**
The Niels Bohr Institute
Blegdamsvej 17
2100 Copenhagen
Denmark

**G. I. Thorbergsson**
Nordita
Blegdamsvej 17
2100 Copenhagen
Denmark

**John A. Hertz**
Nordita
Blegdamsvej 17
2100 Copenhagen
Denmark

## ABSTRACT

We introduce a cost function for learning in feed–forward neural networks which is an explicit function of the internal representation in addition to the weights. The learning problem can then be formulated as two simple perceptrons and a search for internal representations. Back–propagation is recovered as a limit. The frequency of successful solutions is better for this algorithm than for back–propagation when weights and hidden units are updated on the same timescale i.e. once every learning step.

## 1   INTRODUCTION

In their review of back–propagation in layered networks, Rumelhart et al. (1986) describe the learning process in terms of finding good "internal representations" of the input patterns on the hidden units. However, the search for these representations is an indirect one, since the variables which are adjusted in its course are the connection weights, not the activations of the hidden units themselves when specific input patterns are fed into the input layer. Rather, the internal representations are represented implicitly in the connection weight values.

More recently, Grossman et al. (1988 and 1989)[1] suggested a way in which the search for internal representations could be made much more explicit. They proposed to make the activations of the hidden units for each of the input patterns

explicit variables to be adjusted iteratively (together with the weights) in the learning process. However, although they found that the algorithm they gave for making these adjustments could be effective in some test problems, it is rather *ad hoc* and it is difficult to see whether the algorithm will converge to a good solution.

If an optimization task is posed in terms of a cost function which is systematically reduced as the algorithm runs, one is in a much better position to answer questions like these. This is the motivation for this work, where we construct a cost function which is an explicit function of the internal representations as well as the connection weights. Learning is then a descent on the cost function surface, and variations in the algorithm, corresponding to variations in the parameters of the cost function, can be studied systematically. Both the conventional back–propagation algorithm and that of Grossman et al. can be recovered in special limits of ours. It is easy to change the algorithm to include constraints on the learning.

A method somewhat similar to ours has been proposed by Rohwer (1989)[2]. He considers networks with feedback but in this paper we study feed–forward networks. Le Cun has also been working along the same lines, but in a quite different formulation (Le Cun, 1987).

The learning problem for a two–layer perceptron is reduced to learning in two simple perceptrons and the search for internal representations. This search can be carried out by gradient descent of the cost function or by an iterative method.

## 2   THE COST FUNCTION

We work within the standard architecture, with three layers of units and two of connections. Input pattern number $\mu$ is denoted $\xi_k^\mu$, the corresponding target pattern $\zeta_i^\mu$, and its internal representation $\sigma_j^\mu$. We use a convention in which $i$ always labels output units, $j$ labels hidden units, and $k$ labels input units. Thus $w_{ij}$ is always a hidden–to–output weight and $w_{jk}$ an input–to–hidden connection weight. Then the actual activations of the hidden units when pattern $\mu$ is the input are

$$S_j^\mu = g(h_j^\mu) \equiv g(\sum_k w_{jk}\xi_k^\mu) \tag{1}$$

and those of the output units, when given the internal representations $\sigma_j^\mu$ as inputs, are

$$S_i^\mu = g(h_i^\mu) \equiv g(\sum_j w_{ij}\sigma_j^\mu) \tag{2}$$

where $g(h)$ is the activation function, which we take to be $\tanh h$.

The cost function has two terms, one of which describes simple delta–rule learning (Rumelhart et al., 1986) of the internal representations from the inputs by the first layer of connections, and the other of which describes the same kind of learning of the

target patterns from the internal representations in the second layer of connections. We use the "entropic" form for these terms:

$$E = \sum_{i\mu\pm} \tfrac{1}{2}(1 \pm \zeta_i^\mu)\ln\left(\frac{1 \pm \zeta_i^\mu}{1 \pm S_i^\mu}\right) + T\sum_{j\mu\pm} \tfrac{1}{2}(1 \pm \sigma_j^\mu)\ln\left(\frac{1 \pm \sigma_j^\mu}{1 \pm S_j^\mu}\right) \qquad (3)$$

This form of the cost function has been shown to reduce the learning time (Solla et al., 1988). We allow different relative weights for the two terms through the parameter $T$. This cost function should now be minimized with respect to the two sets of connection weights $w_{ij}$ and $w_{jk}$ *and* the internal representations $\sigma_j^\mu$.

The resulting gradient descent learning equations for the connection weights are simply those of simple one–layer perceptrons:

$$\frac{\partial w_{ij}}{\partial t} \propto -\frac{\partial E}{\partial w_{ij}} = \sum_\mu (\zeta_i^\mu - S_i^\mu)\sigma_j^\mu \equiv \sum_\mu \delta_i^\mu \sigma_j^\mu \qquad (4)$$

$$\frac{\partial w_{jk}}{\partial t} \propto -\frac{\partial E}{\partial w_{jk}} = T\sum_\mu (\sigma_j^\mu - S_j^\mu)\xi_k^\mu \equiv T\sum_\mu \delta_j^\mu \xi_k^\mu \qquad (5)$$

The new element is the corresponding equation for the adjustment of the internal representations:

$$\frac{\partial \sigma_j^\mu}{\partial t} \propto -\frac{\partial E}{\partial \sigma_j^\mu} = \sum_i \delta_i^\mu w_{ij} + T h_j^\mu - T\tanh^{-1}\sigma_j^\mu \qquad (6)$$

The stationary values of the internal representations thus solve

$$\sigma_j^\mu = \tanh(h_j^\mu + T^{-1}\sum_i \delta_i^\mu w_{ij}) \qquad (7)$$

which has a simple interpretation: The internal representation variables $\sigma_j^\mu$ are like conventional units except that in addition to the field fed forward into them from the input layer they also feel the back–propagated error field $b_j^\mu \equiv \sum_i \delta_i^\mu w_{ij}$. The parameter $T$ regulates the relative weights of these terms.

Instead of doing gradient descent we have iterated equation (7) to find the internal representations.

One of the advantages of formulating the learning problem in terms of a cost function is that it is easy to implement constraints on the learning. Suppose we want to prevent the network from forming the same internal representations for different output patterns. We can then add the term

$$E = \frac{\gamma}{2}\sum_{ij\mu\nu} \zeta_i^\mu \zeta_i^\nu \sigma_j^\mu \sigma_j^\nu \qquad (8)$$

to the energy. We may also want to suppress internal representations where the units have identical values. This may be seen as an attempt to produce efficient representations. The term

$$E = \frac{\gamma'}{2} \sum_\mu \left( \sum_j \sigma_j^\mu \right)^2 \tag{9}$$

is then added to the energy. The parameters $\gamma$ and $\gamma'$ can be tuned to get the best performance. With these new terms equation (7) for the internal representations becomes

$$\sigma_j^\mu = \tanh(h_j^\mu + T^{-1} \sum_i \delta_i^\mu w_{ij} + \gamma T^{-1} \sum_{i\nu} \zeta_i^\mu \zeta_i^\nu \sigma_j^\nu + \gamma' T^{-1} \sum_{j'} \sigma_{j'}^\mu). \tag{10}$$

The only change in the algorithm is that this equation is iterated rather than (7). These terms lead to better performance in some problems. The benefit of including such terms is very problem–dependent. We include in our results an example where these terms are useful.

## 3   SIMPLE LIMITS

It is simple to recover ordinary back–propagation in this model. It is the limit where $T \gg 1$: Expanding (7) we obtain

$$\sigma_j^\mu = S_j^\mu + T^{-1} \sum_i \delta_i^\mu w_{ij} (1 - \tanh^2 h_j^\mu) \tag{11}$$

Keeping only the lowest–order surviving terms, the learning equations for the connection weights then reduce to

$$\frac{\partial w_{ij}}{\partial t} = \sum_\mu [\zeta_i^\mu - \tanh(\sum_{j'} w_{ij'} S_{j'}^\mu)] S_j^\mu \tag{12}$$

and

$$\frac{\partial w_{jk}}{\partial t} = \sum_{i\mu} \delta_i^\mu w_{ij} (1 - \tanh^2 h_j^\mu) \xi_k^\mu \tag{13}$$

which are just the standard back–propagation equations (with an entropic cost function).

Now consider the opposite limit, $T \ll 1$. Then the second term dominates in (7):

$$\sigma_j^\mu \rightarrow \text{sgn} \left( \sum_i \delta_i^\mu w_{ij} \right) \tag{14}$$

A similar algorithm to the one of Grossman et al. is then to train the input–to–hidden connection weights with these $\sigma_j^\mu$ as targets while training the hidden–to–output weights with the $\sigma_j^\mu$ obtained in the other limit (7) as inputs. That is, one alternates between high and low $T$ according to which layer of weights one is adjusting.

# 4  RESULTS

There are many ways to do the optimization in practice. To be able to make a comparison with back–propagation, we have made simulations that, at high $T$, are essentially the same as back–propagation (in terms of weight adjustment).

In one set of simulations we have kept the internal representations, $\sigma_j^\mu$, optimal with the given set of connections. This means that after one step of weight changes we have relaxed the $\sigma$'s. One can think of the $\sigma$'s as fast–varying and the weights as slowly–varying. In the $T \gg 1$ limit we can use these simulations to get a comparison with back–propagation as described in the previous section.

In our second set of simulations we iterate the equation for the $\sigma$'s only once after one step of weight updating. All variables are then updated on the same timescale. This turns out to increase the success rate for learning considerably compared to the back–propagation limit. The $\sigma$'s are updated in random order such that each one is updated once on the average.

The learning rate, momentum, etc. have been chosen optimally for the back–propagation limit (large $T$) and kept fixed at these values for other values of $T$ (though no systematic optimization of parameters has been done).

We have tested the algorithm on the parity and encoding problems for $T = 1$ and $T = 10$ (the back–propagation limit). Each problem was run 100 times and the average error and success rate were measured and plotted as functions of learning steps (time). One learning step corresponds to one updating of the weights.

For the parity problem (and other similar tasks) the learning did not converge for $T$ lower than about 3. When the weights are small we can expand the tanh on the output in equation (7),

$$\sigma_j^\mu \simeq \tanh(h_j^\mu + T^{-1} \sum_i w_{ij}[\zeta_i^\mu - \sum_{j'} w_{ij'}\sigma_{j'}^\mu]), \tag{15}$$

so the $\sigma_j^\mu$ sits in a spin-glass-like "local field" except for the connection to itself. When the algorithm is started with small random weights this self–coupling $(\sum_i (w_{ij})^2)$ is dominant. Forcing the self–coupling to be small at low $w$'s and gradually increasing it to full strength when the units saturate improves the performance a lot.

For larger networks the self–coupling does not seem to be a problem.

The specific test problems were:

**Parity** with 4 input units and 4 hidden units and all the 16 patterns in the training set. We stop the runs after 300 sweeps of the training set. For $T = 1$ the self coupling is suppressed.

**Encoding** with 8 input, 3 hidden and 8 output units and 8 patterns to learn (same input as output). The 8 patterns have $-1$ at all units but one. We stop the runs after 500 sweeps of the training set.

Both problems were run with fast–varying $\sigma$'s and with all variables updated on the same timescale. We determined the average learning time of the *successful* runs and the percentage of the 100 trials that were successful. The success criterion was that the sign of the output was correct. The learning times and success rates are shown in table 1.

**Table 1:** Learning Times and Succes Rates

|  |  | *Learning times* | | *Success rate* | |
|---|---|---|---|---|---|
|  |  | T=1 | T=10 | T=1 | T=10 |
| Fast-vary- | Parity | 130±10 | 97±6 | 30% | 48% |
| ing $\sigma$'s | Encoding | 167±10 | 88±4 | 95% | 98% |
| Slow-vary- | Parity | 146±10 | 121±6 | 36% | 57% |
| ing $\sigma$'s | Encoding | 145±8 | 64±2 | 99% | 100% |

In figure 1 we plot the average error as a function of learning steps and the success rate for each set of runs.

It can seem a disadvantage of this method that it is necessary to store the values of the $\sigma$'s between learning sweeps. We have therefore tried to start the iteration of equation (7) with the value $\sigma_j^\mu = \tanh(\sum_k w_{jk}\xi_k^\mu)$ on the right hand side. This does not affect the performance much.

We have investigated the effect of including the terms (8) and (9) in the energy. For the same parity problem as above we get an improved success rate in the high $T$ limit.

## 5   CONCLUSION

The most striking result is the improvement in the success rate when all variables, weights and hidden units, are updated once every learning step. This is in contrast to back–propagation, where the values of the hidden units are completely determined by the weights and inputs. In our formulation this corresponds to relaxing the hidden units fully in every learning cycle and having the parameter $T \gg 1$. There is then an advantage in considering the hidden units as additional variables during the learning phase whose values are not completely determined by the field fed forward to them from the inputs.

The results indicate that the performance of the algorithm is best in the high $T$ limit.

For the parity problem the performance of the algorithm presented here is similar to that of the back–propagation algorithm measured in learning time. The real advantage is the higher frequency of successful solutions. For the encoding problem the algorithm is faster than back–propagation but the success rate is similar ($\simeq$ 100%). The algorithm should also be comparable to back–propagation in cpu time

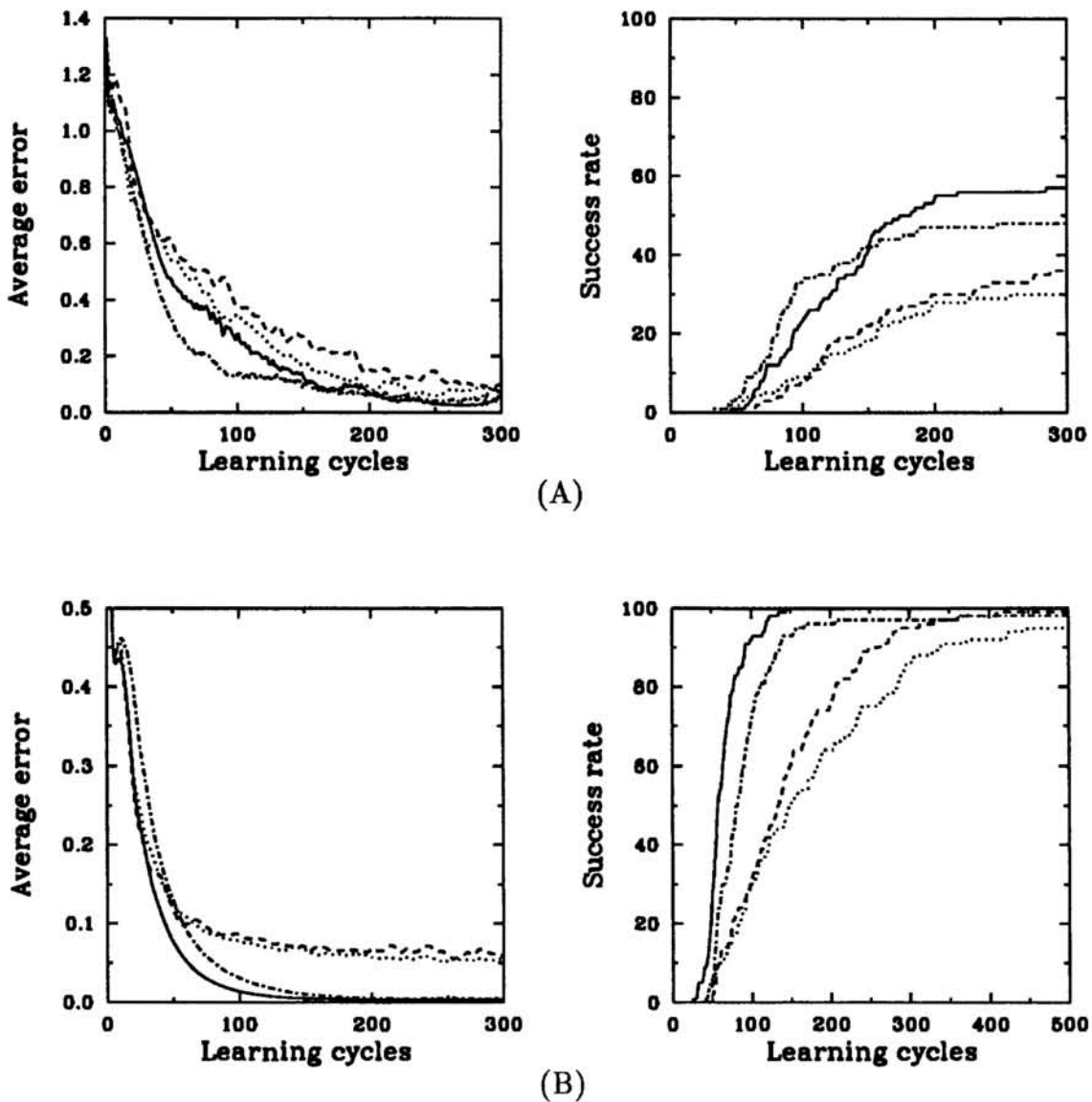

**Figure 1:** (A) The left plot shows the error as a function of learning time for the 4-parity problem for those runs that converged within 300 learning steps. The curves are: $T = 10$ and slow sigmas (———), $T = 10$ and fast sigmas (········), $T = 1$ and slow sigmas (------), and $T = 1$ and fast sigmas (·········). The right plot is the percentage of converged runs as a function of learning time.
(B) The same as above but for the encoding problem.

in the limit where all variables are updated on the same timescale (once every learning sweep).

Because the computational complexity is shifted from the calculation of new weights to the determination of internal representations, it might be easier to implement this method in hardware than back–propagation is. It is possible to use the method *without* saving the array of internal representations by using the field fed forward from the inputs to generate an internal representation that then becomes a starting point for iterating the equation for $\sigma$.

The method can easily be generalized to networks with feedback (as in [Rohwer, 1989]) and it would be interesting to see how it compares to other algorithms for recurrent networks. There are many other directions in which one can continue this work. One is to try another cost function. Another is to use binary units and perceptron learning.

**References**

Le Cun, Y (1987). Modeles Connexionistes de l'Apprentissage. Thesis, Paris.

Grossman, T, R Meir and E Domany (1988). Learning by Choice of Internal Representations. *Complex Systems* **2**, 555.

Grossman, T (1989). The CHIR Algorithm: A Generalization for Multiple Output and Multilayered Networks. Preprint, submitted to *Complex Systems*.

Rohwer, R (1989). The "Moving Targets" Training Method. Preprint, Edinburgh.

Rumelhart, D E, G E Hinton and R J Williams (1986). Chapter 8 in *Parallel Distributed Processing*, vol 1 (D E Rumelhart and J L McClelland, eds), MIT Press.

Solla, S A, E Levin, M Fleisher (1988). Accelerated Learning in Layered Neural Networks. *Complex Systems* **2**, 625.

## Footnotes

[1]See also the paper by Grossman in this volume.

[2]See also the paper by Rohwer in this volume.
